# Modeling Consistency in a Speaker Independent Continuous Speech Recognition System

**Yochai Konig, Nelson Morgan, Chuck Wooters**
International Computer Science Institute
1947 Center Street, Suite 600
Berkeley, CA 94704, USA.

**Victor Abrash, Michael Cohen, Horacio Franco**
SRI International
333 Ravenswood Ave.
Menlo Park, CA 94025, USA

## Abstract

We would like to incorporate speaker-dependent consistencies, such as gender, in an otherwise speaker-independent speech recognition system. In this paper we discuss a Gender Dependent Neural Network (GDNN) which can be tuned for each gender, while sharing most of the speaker independent parameters. We use a classification network to help generate gender-dependent phonetic probabilities for a statistical (HMM) recognition system. The gender classification net predicts the gender with high accuracy, 98.3% on a Resource Management test set. However, the integration of the GDNN into our hybrid HMM-neural network recognizer provided an improvement in the recognition score that is not statistically significant on a Resource Management test set.

## 1 INTRODUCTION

Earlier work [Bourlard and Morgan, 1991] has shown the ability of Multilayer Perceptrons (MLPs) to estimate emission probabilities for Hidden Markov Models (HMM). As shown in their report, with a few assumptions, an MLP may be viewed as estimating the probability $P(q|x)$ where $q$ is a subword model (or a state of a subword model) and $x$ is the input acoustic

speech data. In this hybrid HMM/MLP recognizer, it was shown that these estimates led to improved performance over standard estimation techniques when a fairly simple HMM was used. More recent results have shown improvements using hybrid HMM/MLP probability estimation over a state-of-the-art pure HMM-based system[Cohen *et al.*, 1993; Renals *et al.*, 1992].

Some speaker dependencies exist in common parametric representations of speech, and it is possible that making the dependencies explicit may improve performance for a given speaker (essentially enabling the recognizer to soften the influence of the speaker dependency). The basic problem with modeling and estimating explicitly speaker dependent parameters is the lack of training data. In the limit, the only available information about a new speaker is the utterance to be recognized. This limit is our starting point for this study. Even with this limited information, we can incorporate constraints on analysis that rely on the same speaker producing all the frames in an utterance, thus ensuring consistency. As has been observed for some mainstream Hidden Markov Models (HMM) systems [Murveit *et al.*, 1990], given enough training data, separate phonetic models for male and female speakers can be used to improve performance. Our first attack on consistency, then, is to incorporate gender consistency in the recognition process. In contrast to non-connectionist HMM systems, our proposed architecture attempts to share the gender-independent parameters.

Our study had two steps: first we trained an MLP to estimate the probability of gender. Then, we investigated ways to integrate the gender consistency constraint into our existing MLP-HMM hybrid recognizer, resulting in our GDNN architecture. We will give a short description of some related work, followed by an explanation of the two steps described above. We conclude with some discussion and thoughts about future work.

## 2   RELATED AND PREVIOUS WORK

Our previous experiments with the Gender-Dependent Neural Network (GDNN) are described in [Abrash *et al.*, 1992; Konig and Morgan, 1992]. Other researchers have worked on related problems. For example Hampshire and Waibel presented the "Meta-Pi" architecture [Hampshire and Waibel, 1990]. The building blocks for the "Meta-Pi" architecture are multiple TDNN's that are trained to recognize the speech of an individual speaker. These building blocks are integrated by another multiple TDNN trained in a Bayesian MAP scheme to maximize the phoneme recognition rate of the overall architecture. The performance of the "Meta-Pi" architecture on a six speaker /b,d,g/ task was comparable to a speaker dependent system on the same task.

Another example of related work is speaker normalization, which attempts to minimize between-speaker variations by transforming the data of a new speaker to that of a reference speaker, and then applying the speaker dependent system for the reference speaker [Huang *et al.*, 1991].

## 3   THE CLASSIFICATION NET

In order to classify the gender of a new speaker we need features that distinguish between speakers, in contrast to the features that are used for phoneme recognition that are chosen to suppress speaker variations. Given our constraint that the only available information

about the new speaker is the sentence to be recognized, we chose features that are a rough estimate of the vocal tract properties and the fundamental frequency of the new speaker. Furthermore, we tried to suppress the linguistic information in our estimate. More specifically, the goal was to build a net that estimates the probability $P(Gender|Data)$. After some experimentation, the first twelve LPC cepstral coefficients were calculated over a 20 msec window every 10 msec (50% overlap) and averaged along each sentence. The sampling rate was 16khz. These features were augmented by an estimate of the fundamental frequency for a total of 13 features per sentence. The MLP had one hidden layer with 24 hidden units. There were two output units, one for each gender. The training set was the 109-speaker DARPA Resource Management corpus. 3500 sentences were used for the training set and 490 in the cross validation set. The size of the test set was 600 sentences, and it was a combination of the DARPA Resource Management speaker-independent Feb89 and Oct89 test sets. The trained MLP predicts the gender for the test set with less than 1.7% error on the sentence level.

## 4   INCORPORATING GENDER CONSISTENCY INTO OUR HYBRID HMM/MLP RECOGNIZER

### 4.1   DISCUSSION

Our goal is to find an architecture that shares the gender independent parameters and models the gender dependent parameters. Given our gender consistency constraint we estimate a probability that is explicitly conditioned on gender, as if the phonetic models were simply doubled to permit male and female forms of each phoneme. We can express $P(male, phone|data)$ (which is then divided by priors to get the corresponding data likelihood) by expansion to $P(phone|male, data) \times P(male|data)$. This factorization is realized by two separate MLP's: $P(male|data)$ is estimated by the classification net described above, and $P(phone|male, data)$ is realized by our GDNN described below. For further description on how to factorize probabilities by neural networks see [Morgan and Bourlard, 1992]. The final likelihood for the male case can be expressed as:

$$P(data|phone, male) = \frac{P(phone|male, data) \times P(male|data) \times P(data)}{P(phone|male) \times P(male)} \quad (1)$$

Note that during recognition, $P(data)$ can be ignored. Similarly, a female-assumed probability can be computed for each hypothesized phone. These male and female-assumed probabilities can then be used in separate Viterbi calculations (since we do not permit any hypothesis to switch gender in the midst of an utterance). In other words, dynamic programming is used with the framewise network outputs to evaluate the best hypothesized utterance assuming male gender, and then the same is done for the female case. The case with the lowest cost (highest probability) is then chosen. Note that the output of the classification net only helps in choosing between the sentence recognized according to female gender or male gender.

The critical question is how to estimate $P(phone|gender, data)$. A possible answer is to have two separate nets, one trained only on males, and the other trained only on females. This approach has the potential disadvantages of doubling the number of parameters in the system, and of not sharing the gender independent parameters. We have experimented with a such a net [Konig and Morgan, 1992] and it improved our result over the baseline system.

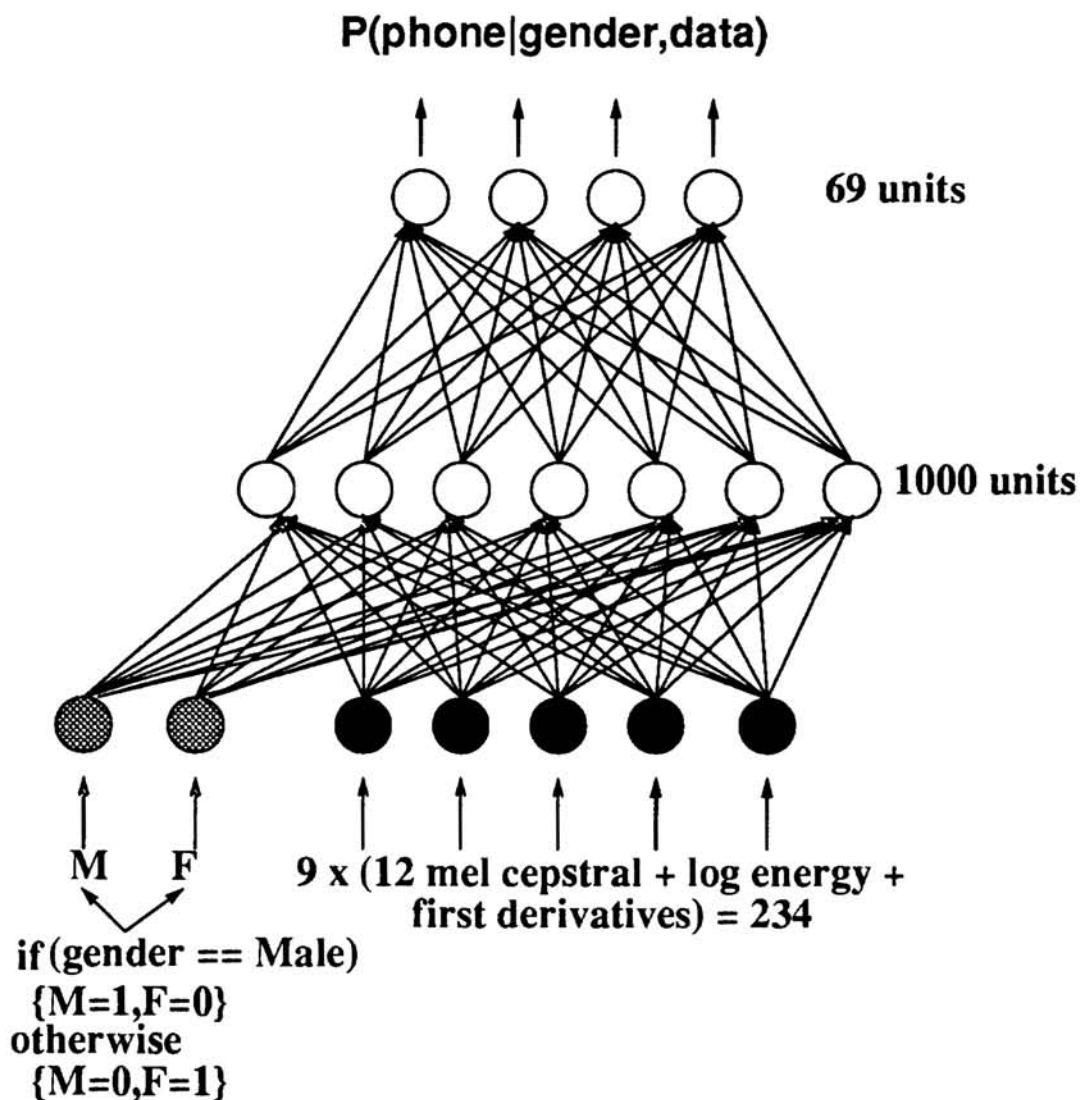

**P(phone|gender,data)**

69 units

1000 units

M    F

9 x (12 mel cepstral + log energy +
first derivatives) = 234

if (gender == Male)
  {M=1,F=0}
otherwise
  {M=0,F=1}

Figure 1: **A Gender Dependent Neural Network(GDNN)**

We present here a hybrid GDNN architecture that has the flexibility to tune itself to each gender. The idea is to have extra binary inputs that specify the gender of the speaker and then the probabilities that the network estimates will be conditioned on the gender. The architecture is shown in figure 1.

## 4.2  EXPERIMENTS AND RESULTS

We have compared four different architectures. The first architecture is our baseline system, namely, one large net that was trained on all the sentences in the training set. The second uses two separate nets, for males and females. The third is the hybrid GDNN architecture described in figure 1. The fourth architecture is a variant of the third architecture, the difference being that the binary units are connected to the output units instead of to the hidden units. All the nets have 1000 hidden units and 69 output units, including the totally separate male and female nets. While one might think that the consequent doubling of

Table 1: Result Summary

| Architecture | Test Set Word Error |
|---|---|
| Baseline | 10.6% |
| Two Separate Nets | 10.9% |
| Hybrid Architecture - Variant(Binary to Output) | 10.9% |
| GDNN Binary to hidden | 10.2% |

the number of parameters in the system might explain the observed degradation for the performance for the second architecture, we have also experimented with several sizes of male and female separate nets, by changing the number of the hidden units and the number of input frames. None of these experiments resulted in a significant improvement. We used 12 mel-cepstral features and the log-energy along with their first derivatives, so the number of input features per frame was 26. The features were calculated from 20ms of speech, computed every 10 msec (as before). The length of the temporal window (the number of input frames) was 9, so the total number of input features was 234. The training set was the 109-speaker DARPA Resource Management corpus. 3500 sentences were used for the training set and 490 in the cross validation set. The size of the test was the 600 sentences making up the DARPA Feb89 and Oct89 test sets. The results are summarized in table 1, and are achieved using the standard Resource Management wordpair grammar(perplexity = 60) with a simple context-independent HMM recognizer. We should note here that these results are all somewhat worse than our other results published in [Renals et al., 1992; Cohen et al., 1993], as the latter were achieved using SRI's phonological models, and these were done with a single-pronunciation single-state HMM (with each state repeated for a rough duration model).

## 5 DISCUSSION AND FUTURE WORK

The best results were achieved by the GDNN hybrid architecture that shares the gender-independent parameters while modeling the gender dependent parameters. However the improvement over our baseline is not statistically significant for this test set, although it is consistent with our experiments with other test sets, not reported here. A possible source for further improvement is using a training set with a more balanced representation of gender. In the DARPA Resource Management speaker independent training set there are 2830 sentences uttered by males versus only 1160 sentences uttered by females. Thus, performance may have suffered from an insufficient number of female training sentences. A reasonable extension to this work would be the modeling of additional speaker dependent parameters such as speech rate, accent, etc. Another direction that might be more fruitful is to combine the gender-dependent models in the local estimation of phonemes, and not to do separate Viterbi recognitions for each gender. We are currently examining this latter alternative.

**Acknowledgements**

Thanks to Steve Renals for his comments along the way. Computations were done on the RAP machine, with support from software guru Phil Kohn, and hardware wiz Jim Beck. Thanks to Hynek Hermansky for advising us about the features for the gender classification net. Thanks to the other members of the speech group at ICSI for their helpful comments. This work was partially funded by DARPA contract MDA904-90-C-5253.

## References

[Abrash *et al.*, 1992] V. Abrash, H. Franco, M. Cohen, N. Morgan, and Y. Konig. Connectionist gender adaptation in a hybrid neural network / hidden markov model speech recognition system. In *Proc. Int'l Conf. on Spoken Lang. Processing*, Banff, Canada, October 1992.

[Bourlard and Morgan, 1991] H. Bourlard and N. Morgan. Merging multilayer perceptrons & hidden markov models: Some experiments in continuous speech recognition. In E. Gelenbe, editor, *Artificial Neural Networks: Advances and Applications*. North Holland Press, 1991.

[Cohen *et al.*, 1993] M. Cohen, H. Franco, N. Morgan, D. Rumelhart, and V. Abrash. Context-dependent multiple distribution phonetic modeling. In C.L. Giles, Hanson S.J, and J.D. Cowan, editors, *Advances in Neural Information Processing Systems*, volume 5. Morgan Kaufmann, San Mateo, 1993.

[Hampshire and Waibel, 1990] J.B. Hampshire and A. Waibel. Connectionist architectures for multi-speaker phoneme recognition. In D.S. Touretzky, editor, *Advances in Neural Information Processing Systems 2*, San mateo, CA, 1990. Morgan Kaufman.

[Huang *et al.*, 1991] X.D. Huang, K.F. Lee, and A. Waibel. Connectionist speaker normalization and its application to speech recognition. In *Neural Networks for Siganl Processing, proc. of 1991 IEEE Workshop,*, Princeton, New Jersey, October 1991.

[Konig and Morgan, 1992] Y. Konig and N. Morgan. Gdnn: A gender -dependent neural network for continuous speech recognition. In *Proc. international Joint Conference on Neural Networks*, Baltimore, Maryland, June 1992.

[Morgan and Bourlard, 1992] N. Morgan and H. Bourlard. Factoring neural networks by a statistical method. *Neural Computation*, (4):835–838, 1992.

[Murveit *et al.*, 1990] H. Murveit, M. Weintraub, and M. Cohen. Training set issues in sri's decipher speech recognition system. In *Proc. speech and Natural Language Workshop*, pages 337–340, June 1990.

[Renals *et al.*, 1992] S. Renals, N. Morgan, M. Cohen, H. Franco, and H. Bourlard. Connectionist probability estimation in the decipher speech recognition system. In *Proceedings IEEE Intl. Conf. on Acoustics, Speech, and Signal Processing*, San Francisco, California, March 1992. IEEE.